# Sparse Representation for Signal Classification

**Ke Huang and Selin Aviyente**
Department of Electrical and Computer Engineering
Michigan State University, East Lansing, MI 48824
{kehuang, aviyente}@egr.msu.edu

## Abstract

In this paper, application of sparse representation (factorization) of signals over an overcomplete basis (dictionary) for signal classification is discussed. Searching for the sparse representation of a signal over an overcomplete dictionary is achieved by optimizing an objective function that includes two terms: one that measures the signal reconstruction error and another that measures the sparsity. This objective function works well in applications where signals need to be reconstructed, like coding and denoising. On the other hand, discriminative methods, such as linear discriminative analysis (LDA), are better suited for classification tasks. However, discriminative methods are usually sensitive to corruption in signals due to lacking crucial properties for signal reconstruction. In this paper, we present a theoretical framework for signal classification with sparse representation. The approach combines the discrimination power of the discriminative methods with the reconstruction property and the sparsity of the sparse representation that enables one to deal with signal corruptions: noise, missing data and outliers. The proposed approach is therefore capable of robust classification with a sparse representation of signals. The theoretical results are demonstrated with signal classification tasks, showing that the proposed approach outperforms the standard discriminative methods and the standard sparse representation in the case of corrupted signals.

## 1 Introduction

Sparse representations of signals have received a great deal of attentions in recent years. The problem solved by the sparse representation is to search for the most compact representation of a signal in terms of linear combination of atoms in an overcomplete dictionary. Recent developments in multi-scale and multi-orientation representation of signals, such as wavelet, ridgelet, curvelet and contourlet transforms are an important incentive for the research on the sparse representation. Compared to methods based on orthonormal transforms or direct time domain processing, sparse representation usually offers better performance with its capacity for efficient signal modelling. Research has focused on three aspects of the sparse representation: pursuit methods for solving the optimization problem, such as matching pursuit [1], orthogonal matching pursuit [2], basis pursuit [3], LARS/homotopy methods [4]; design of the dictionary, such as the K-SVD method [5]; the applications of the sparse representation for different tasks, such as signal separation, denoising, coding, image inpainting [6, 7, 8, 9, 10]. For instance, in [6], sparse representation is used for image separation. The overcomplete dictionary is generated by combining multiple standard transforms, including curvelet transform, ridgelet transform and discrete cosine transform. In [7], application of the sparse representation to blind source separation is discussed and experimental results on EEG data analysis are demonstrated. In [8], a sparse image coding method with the wavelet transform is presented. In [9], sparse representation with an adaptive dictionary is shown to have state-of-the-art performance in image denoising. The widely used shrinkage method for image desnoising is shown to be the first iteration of basis pursuit that solves the sparse representation problem [10].

In the standard framework of sparse representation, the objective is to reduce the signal reconstruction error with as few number of atoms as possible. On the other hand, discriminative analysis methods, such as LDA, are more suitable for the tasks of classification. However, discriminative methods are usually sensitive to corruption in signals due to lacking crucial properties for signal reconstruction. In this paper, we propose the method of sparse representation for signal classification (SRSC), which modifies the standard sparse representation framework for signal classification. We first show that replacing the reconstruction error with discrimination power in the objective function of the sparse representation is more suitable for the tasks of classification. When the signal is corrupted, the discriminative methods may fail because little information is contained in discriminative analysis to successfully deal with noise, missing data and outliers. To address this robustness problem, the proposed approach of SRSC combines discrimination power, signal reconstruction and sparsity in the objective function for classification. With the theoretical framework of SRSC, our objective is to achieve a sparse and robust representation of corrupted signals for effective classification.

The rest of this paper is organized as follows. Section 2 reviews the problem formulation and solution for the standard sparse representation. Section 3 discusses the motivations for proposing SRSC by analyzing the reconstructive methods and discriminative methods for signal classification. The formulation and solution of SRSC are presented in Section 4. Experimental results with synthetic and real data are shown in Section 5 and Section 6 concludes the paper with a summary of the proposed work and discussions.

## 2 Sparse Representation of Signal

The problem of finding the sparse representation of a signal in a given overcomplete dictionary can be formulated as follows. Given a $N \times M$ matrix $\mathbf{A}$ containing the elements of an overcomplete dictionary in its columns, with $M > N$ and usually $M >> N$, and a signal $\mathbf{y} \in R^N$, the problem of sparse representation is to find an $M \times 1$ coefficient vector $\mathbf{x}$, such that $\mathbf{y} = \mathbf{A}\mathbf{x}$ and $\|\mathbf{x}\|_0$ is minimized, i.e.,

$$\mathbf{x} = \min_{\mathbf{x}'} \|\mathbf{x}'\|_0 \quad \text{s.t.} \quad \mathbf{y} = \mathbf{A}\mathbf{x}. \tag{1}$$

where $\|\mathbf{x}\|_0$ is the $\ell_0$ norm and is equivalent to the number of non-zero components in the vector $\mathbf{x}$. Finding the solution to equation (1) is NP hard due to its nature of combinational optimization. Suboptimal solutions to this problem can be found by iterative methods like the matching pursuit and orthogonal matching pursuit. An approximate solution is obtained by replacing the $\ell_0$ norm in equation (1) with the $\ell_1$ norm, as follows:

$$\mathbf{x} = \min_{\mathbf{x}'} \|\mathbf{x}'\|_1 \quad \text{s.t.} \quad \mathbf{y} = \mathbf{A}\mathbf{x}. \tag{2}$$

where $\|\mathbf{x}\|_1$ is the $\ell_1$ norm. In [11], it is proved that if certain conditions on the sparsity is satisfied, i.e., the solution is sparse enough, the solution of equation (1) is equivalent to the solution of equation (2), which can be efficiently solved by basis pursuit using linear programming. A generalized version of equation (2), which allows for certain degree of noise, is to find $\mathbf{x}$ such that the following objective function is minimized:

$$J_1(\mathbf{x}; \lambda) = \|\mathbf{y} - \mathbf{A}\mathbf{x}\|_2^2 + \lambda \|\mathbf{x}\|_1 \tag{3}$$

where the parameter $\lambda > 0$ is a scalar regularization parameter that balances the tradeoff between reconstruction error and sparsity. In [12], a Bayesian approach is proposed for learning the optimal value for $\lambda$. Except for the intuitive interpretation as obtaining a sparse factorization that minimizes signal reconstruction error, the problem formulated in equation (3) has an equivalent interpretation in the framework of Bayesian decision as follows [13]. The signal $\mathbf{y}$ is assumed to be generated by the following model:

$$\mathbf{y} = \mathbf{A}\mathbf{x} + \varepsilon \tag{4}$$

where $\varepsilon$ is white Gaussian noise. Moreover, the prior distribution of $\mathbf{x}$ is assumed to be super-Gaussian:

$$p(\mathbf{x}) \sim \exp\left(-\lambda \sum_{i=1}^{M} |x_i|^p\right) \tag{5}$$

where $p \in [0, 1]$. This prior has been shown to encourage sparsity in many situations, due to its heavy tails and sharp peak. Given this prior, maximum *a posteriori* (MAP) estimation of $\mathbf{x}$ is formulated as

$$\mathbf{x}_{MAP} = \arg\max_{\mathbf{x}} p(\mathbf{x}|\mathbf{y}) = \arg\min_{\mathbf{x}}[-\log p(\mathbf{y}|\mathbf{x}) - \log p(\mathbf{x})] = \arg\min_{\mathbf{x}}(\|\mathbf{y} - \mathbf{A}\mathbf{x}\|_2^2 + \lambda \|\mathbf{x}\|_p)$$
(6)

when $p = 0$, equation (6) is equivalent to the generalized form of equation (1); when $p = 1$, equation (6) is equivalent to equation (2).

# 3 Reconstruction and Discrimination

Sparse representation works well in applications where the original signal $\mathbf{y}$ needs to be reconstructed as accurately as possible, such as denoising, image inpainting and coding. However, for applications like signal classification, it is more important that the representation is discriminative for the given signal classes than a small reconstruction error.

The difference between reconstruction and discrimination has been widely investigated in literature. It is known that typical *reconstructive* methods, such as principal component analysis (PCA) and independent component analysis (ICA), aim at obtaining a representation that enables sufficient reconstruction, thus are able to deal with signal corruption, i.e., noise, missing data and outliers. On the other hand, *discriminative* methods, such as LDA [14], generate a signal representation that maximizes the separation of distributions of signals from different classes. While both methods have broad applications in classification, the discriminative methods have often outperformed the reconstructive methods for the classification task [15, 16]. However, this comparison between the two types of method assumes that the signals being classified are ideal, i.e., noiseless, complete(without missing data) and without outliers. When this assumption does not hold, the classification may suffer from the nonrobust nature of the discriminative methods that contains insufficient information to successfully deal with signal corruptions. Specifically, the representation provided by the discriminative methods for optimal classification does not necessarily contain sufficient information for signal reconstruction, which is necessary for removing noise, recovering missing data and detecting outliers. This performance degradation of discriminative methods on corrupted signals is evident in the examples shown in [17]. On the other hand, reconstructive methods have shown successful performance in addressing these problems. In [9], the sparse representation is shown to achieve state-of-the-art performance in image denoising. In [18], missing pixels in images are successfully recovered by inpainting method based on sparse representation. In [17, 19], PCA method with subsampling effectively detects and excludes outliers for the following LDA analysis.

All of these examples motivate the design of a new signal representation that combines the advantages of both reconstructive and discriminative methods to address the problem of *robust classification* when the obtained signals are corrupted. The proposed method should generate a representation that contain discriminative information for classification, crucial information for signal reconstruction and preferably the representation should be sparse. Due to the evident reconstructive properties [9, 18], the available efficient pursuit methods and the sparsity of representation, we choose the sparse representation as the basic framework for the SRSC and incorporate a measure of discrimination power into the objective function. Therefore, the sparse representation obtained by the proposed SRSC contains both crucial information for reconstruction and discriminative information for classification, which enable a reasonable classification performance in the case of corrupted signals. The three objectives: sparsity, reconstruction and discrimination may not always be consistent. Therefore, weighting factors are introduced to adjust the tradeoff among these objectives, as the weighting factor $\lambda$ in equation (3). It should be noted that the aim of SRSC is not to improve the standard discriminative methods like LDA in the case of ideal signals, but to achieve comparable classification results when the signals are corrupted. A recent work [17] that aims at robust classification shares some common ideas with the proposed SRSC. In [17], PCA with subsampling proposed in [19] is applied to detect and exclude outliers in images and the rest of pixels are used for calculating LDA.

# 4 Sparse Representation for Signal Classification

In this section, the SRSC problem is formulated mathematically and a pursuit method is proposed to optimize the objective function. We first replace the term measuring reconstruction error with a term measuring discrimination power to show the different effects of reconstruction and discrimination. Further, we incorporate measure of discrimination power in the framework of standard sparse representation to effectively address the problem of classifying corrupted signals. The Fisher's discrimination criterion [14] used in the LDA is applied to quantify the discrimination power. Other well-known discrimination criteria can easily be substituted.

## 4.1 Problem Formulation

Given $\mathbf{y} = \mathbf{A}\mathbf{x}$ as discussed in Section 2, we view $\mathbf{x}$ as the feature extracted from signal $\mathbf{y}$ for classification. The extracted feature should be as discriminative as possible between the different signal classes. Suppose that we have a set of $K$ signals in a signal matrix $\mathbf{Y} = [\mathbf{y}_1, \mathbf{y}_2, ..., \mathbf{y}_K]$ with the corresponding representation in the overcomplete dictionary as $\mathbf{X} = [\mathbf{x}_1, \mathbf{x}_2, ..., \mathbf{x}_K]$, of which $K_i$ samples are in the class $C_i$, for $1 \leq i \leq C$. Mean $\mathbf{m_i}$ and variance $s_i^2$ for class $C_i$ are computed in the feature space as follows:

$$\mathbf{m_i} = \frac{1}{K_i} \sum_{\mathbf{x} \in C_i} \mathbf{x} \ , \quad s_i^2 = \frac{1}{K_i} \sum_{\mathbf{x} \in C_i} \|\mathbf{x} - \mathbf{m_i}\|_2^2 \tag{7}$$

The mean of all samples are defined as: $\mathbf{m} = \frac{1}{K} \sum_{i=1}^{K} \mathbf{x}_i$. Finally, the Fisher's discrimination power can then be defined as:

$$F(\mathbf{X}) = \frac{S_B}{S_W} = \frac{\left\| \sum_{i=1}^{C} K_i (\mathbf{m}_i - \mathbf{m})(\mathbf{m}_i - \mathbf{m})^T \right\|_2^2}{\sum_{i=1}^{C} s_i^2}. \tag{8}$$

The difference between the sample means $S_B = \left\| \sum_{i=1}^{C} K_i (\mathbf{m}_i - \mathbf{m})(\mathbf{m}_i - \mathbf{m})^T \right\|_2^2$ can be interpreted as the 'inter-class distance' and the sum of variance $S_W = \sum_{i=1}^{C} s_i^2$ can be similarly interpreted as the 'inner-class scatter'. Fisher's criterion is motivated by the intuitive idea that the discrimination power is maximized when the spatial distribution of different classes are as far away as possible and the spatial distribution of samples from the same class are as close as possible.

Replacing the reconstruction error with the discrimination power, the objective function that focuses only on classification can be written as:

$$J_2(\mathbf{X}, \lambda) = F(\mathbf{X}) - \lambda \sum_{i=1}^{K} \|\mathbf{x}_i\|_0 \tag{9}$$

where $\lambda$ is a positive scalar weighting factor chosen to adjust the tradeoff between discrimination power and sparsity. Maximizing $J_2(\mathbf{X}, \lambda)$ generates a sparse representation that has a good discrimination power. When the discrimination power, reconstruction error and sparsity are combined, the objective function can be written as:

$$J_3(\mathbf{X}, \lambda_1, \lambda_2) = F(\mathbf{X}) - \lambda_1 \sum_{i=1}^{K} \|\mathbf{x}_i\|_0 - \lambda_2 \sum_{i=1}^{K} \|\mathbf{y}_i - \mathbf{A}\mathbf{x}_i\|_2^2 \tag{10}$$

where $\lambda_1$ and $\lambda_2$ are positive scalar weighting factors chosen to adjust the tradeoff between the discrimination power, sparsity and the reconstruction error. Maximizing $J_3(\mathbf{X}, \lambda_1, \lambda_2)$ ensures that

a representation with discrimination power, reconstruction property and sparsity is extracted for robust classification of corrupted signals. In the case that the signals are corrupted, the two terms $\sum_{i=1}^{K} \|\mathbf{x}_i\|_0$ and $\sum_{i=1}^{K} \|\mathbf{y}_i - \mathbf{A}\mathbf{x}_i\|_2^2$ robustly recover the signal structure, as in [9, 18]. On the other hand, the inclusion of the term $F(\mathbf{X})$ requires that the obtained representation contains discriminative information for classification. In the following discussions, we refer to the solution of the objective function $J_3(\mathbf{X}, \lambda_1, \lambda_2)$ as the features for the proposed SRSC.

## 4.2 Problem Solution

Both the objective function $J_2(\mathbf{X}, \lambda)$ defined in equation (9) and the objective function $J_3(\mathbf{X}, \lambda_1, \lambda_2)$ defined in equation (10) have similar forms to the objective function defined in the standard sparse representation, as $J_1(\mathbf{x}; \lambda)$ in equation (3). However, the key difference is that the evaluation of $F(\mathbf{X})$ in $J_2(\mathbf{X}, \lambda)$ and $J_3(\mathbf{X}, \lambda_1, \lambda_2)$ involves not only a single sample, as in $J_1(\mathbf{x}; \lambda)$, but also all other samples. Therefore, not all the pursuit methods, such as basis pursuit and LARS/Homotopy methods, that are applicable to the standard sparse representation method can be directly applied to optimize $J_2(\mathbf{X}, \lambda)$ and $J_3(\mathbf{X}, \lambda_1, \lambda_2)$. However, the iterative optimization methods employed in the matching pursuit and the orthogonal matching pursuit provide a direct reference to the optimization of $J_2(\mathbf{X}, \lambda)$ and $J_3(\mathbf{X}, \lambda_1, \lambda_2)$. In this paper, we propose an algorithm similar to the orthogonal matching pursuit and inspired by the simultaneous sparse approximation algorithm described in [20, 21]. Taking the optimization of $J_3(\mathbf{X}, \lambda_1, \lambda_2)$ as example, the pursuit algorithm can be summarized as follows:

1. Initialize the residue matrix $\mathbf{R}_0 = \mathbf{Y}$ and the iteration counter $t = 0$.

2. Choose the atom from the dictionary, $\mathbf{A}$, that maximizes the objective function:

$$\mathbf{g} = \text{argmax}_{\mathbf{g} \in \mathbf{A}} J_3(\mathbf{g}^T \mathbf{R_t}, \lambda_1, \lambda_2) \tag{11}$$

3. Determine the orthogonal projection matrix $\mathbf{P}_t$ onto the span of the chosen atoms, and compute the new residue.

$$\mathbf{R}_t = \mathbf{Y} - \mathbf{P}_t \mathbf{Y} \tag{12}$$

4. Increment $t$ and return to Step 2 until $t$ is less than or equal to a pre-determined number.

The pursuit algorithm for optimizing $J_2(\mathbf{X}, \lambda)$ also follows the same steps. Detailed analysis of this pursuit algorithm can be found in [20, 21].

## 5 Experiments

Two sets of experiments are conducted. In Section 5.1, synthesized signals are generated to show the difference between the features extracted by $J_1(\mathbf{X}, \lambda)$ and $J_2(\mathbf{X}, \lambda)$, which reflects the properties of reconstruction and discrimination. In Section 5.2, classification on real data is shown. Random noise and occlusion are added to the original signals to test the robustness of SRSC.

## 5.1 Synthetic Example

Two simple signal classes, $f_1(t)$ and $f_2(t)$, are constructed with the Fourier basis. The signals are constructed to show the difference between the reconstructive methods and discriminative methods.

$$f_1(t) = g_1 \cos t + h_1 \sin t \tag{13}$$

$$f_2(t) = g_2 \cos t + h_2 \sin t \tag{14}$$

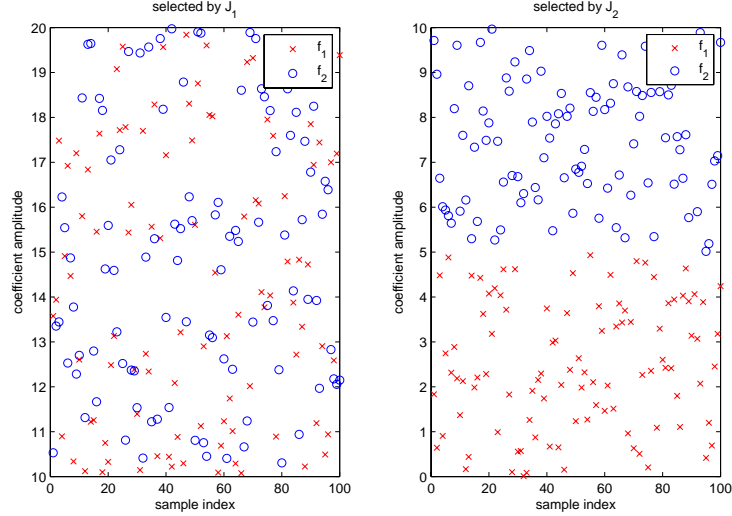

Figure 1: Distributions of projection of signals from two classes with the first atom selected by: $J_1(\mathbf{X}, \lambda)$ (the left figure) and $J_2(\mathbf{X}, \lambda)$ (the right figure).

The scalar $g_1$ is uniformly distributed in the interval $[0, 5]$, and the scalar $g_2$ is uniformly distributed in the interval $[5, 10]$. The scalar $h_1$ and $h_2$ are uniformly distributed in the interval $[10, 20]$. Therefore, most of the energy of the signal can be described by the sine function and most of the discrimination power is in the cosine function. The signal component with most energy is not necessary the component with the most discrimination power. Construct a dictionary as $\{\sin t, \cos t\}$, optimizing the objective function $J_1(\mathbf{X}, \lambda)$ with the pursuit method described in Section 4.2 selects $\sin t$ as the first atom. On the other hand, optimizing the objective function $J_2(\mathbf{X}, \lambda)$ selects $\cos t$ as the first atom. In the simulation, 100 samples are generated for each class and the pursuit algorithm stops at the first run. The projection of the signals from both classes to the first atom selected by $J_1(\mathbf{X}, \lambda)$ and $J_2(\mathbf{X}, \lambda)$ are shown in Fig.1. The difference shown in the figures has direct impact on the classification.

## 5.2 Real Example

Classification with $J_1$, $J_2$ and $J_3$(SRSC) is also conducted on the database of USPS handwritten digits [22]. The database contains 8-bit grayscale images of "0" through "9" with a size of $16 \times 16$ and there are 1100 examples of each digit. Following the conclusion of [23], 10-fold stratified cross validation is adopted. Classification is conducted with the decomposition coefficients (' $\mathbf{X}$' in equation (10)) as feature and support vector machine (SVM) as classifier. In this implementation, the overcomplete dictionary is a combination of Haar wavelet basis and Gabor basis. Haar basis is good at modelling discontinuities in signal and on the other hand, Gabor basis is good at modelling continuous signal components.

In this experiment, noise and occlusion are added to the signals to test the robustness of SRSC. First, white Gaussian noise with increasing level of energy, thus decreasing level of signal-to-noise ratio (SNR), are added to each image. Table 1 summarizes the classification error rates obtained with different SNR. Second, different sizes of black squares are overlayed on each image at a random location to generate occlusion (missing data). For the image size of $16 \times 16$, black squares with size of $3 \times 3$, $5 \times 5$, $7 \times 7$, $9 \times 9$ and $11 \times 11$ are overlayed as occlusion. Table 2 summarizes the classification error rates obtained with occlusion.

Results in Table 1 and Table 2 show that in the case that signals are ideal (without missing data and noiseless) or nearly ideal, $J_2(\mathbf{X}, \lambda)$ is the best criterion for classification. This is consistent with the known conclusion that discriminative methods outperform reconstructive methods in classification. However, when the noise is increased or more data is missing (with larger area of occlusion), the accuracy based on $J_2(\mathbf{X}, \lambda)$ degrades faster than the accuracy base on $J_1(\mathbf{X}, \lambda)$. This indicates

Table 1: Classification error rates with different levels of white Gaussian noise

|  | $Noiseless$ | $20db$ | $15db$ | $10db$ | $5db$ |
|---|---|---|---|---|---|
| $J_1(Reconstruction)$ | 0.0855 | 0.0975 | 0.1375 | 0.1895 | 0.2310 |
| $J_2(Discrimination)$ | 0.0605 | 0.0816 | 0.1475 | 0.2065 | 0.2785 |
| $J_3(SRSC)$ | 0.0727 | 0.0803 | 0.1025 | 0.1490 | 0.2060 |

Table 2: Classification error rates with different sizes of occlusion

|  | no occlusion | $3 \times 3$ | $5 \times 5$ | $7 \times 7$ | $9 \times 9$ | $11 \times 11$ |
|---|---|---|---|---|---|---|
| $J_1(Reconstruction)$ | 0.0855 | 0.0930 | 0.1270 | 0.1605 | 0.2020 | 0.2990 |
| $J_2(Discrimination)$ | 0.0605 | 0.0720 | 0.1095 | 0.1805 | 0.2405 | 0.3305 |
| $J_3(SRSC)$ | 0.0727 | 0.0775 | 0.1135 | 0.1465 | 0.1815 | 0.2590 |

that the signal structures recovered by the standard sparse representation are more robust to noise and occlusion, thus yield less performance degradation. On the other hand, the SRSC demonstrates lower error rate by the combination of the reconstruction property and the discrimination power in the case that signals are noisy or with occlusions.

## 6 Discussions

In summary, sparse representation for signal classification(SRSC) is proposed. SRSC is motivated by the ongoing researches in the area of sparse representation in the signal processing area. SRSC incorporates reconstruction properties, discrimination power and sparsity for robust classification. In current implementation of SRSC, the weight factors are empirically set to optimize the performance. Approaches to determine optimal values for the weighting factors are being conducted, following the methods similar to that introduced in [12].

It is interesting to compare SRSC with the relevance vector machine (RVM) [24]. RVM has shown comparable performance to the widely used support vector machine (SVM), but with a substantially less number of relevance/support vectors. Both SRSC and RVM incorporate sparsity and reconstruction error into consideration. For SRSC, the two terms are explicitly included into objective function. For RVM, the two terms are included in the Bayesian formula. In RVM, the "dictionary" used for signal representation is the collection of values from the "kernel function". On the other hand, SRSC roots in the standard sparse representation and recent developments of harmonic analysis, such as curvelet, bandlet, contourlet transforms that show excellent properties in signal modelling. It would be interesting to see how RVM works by replacing the kernel functions with these harmonic transforms. Another difference between SRSC and RVM is how the discrimination power is incorporated. The nature of RVM is function regression. When used for classification, RVM simply changes the target function value to class membership. For SRSC, the discrimination power is explicitly incorporated by inclusion of a measure based on the Fisher's discrimination. The adjustment of weighting factor in SRSC (in equation (10)) may give some flexibility for the algorithm when facing various noise levels in the signals. A thorough and systemic study of connections and difference between SRSC and RVM would be an interesting topic for the future research.

## References

[1] S. Mallat and Z. Zhang, "Matching pursuits with time-frequency dictionaries," *IEEE Trans. on Signal Processing*, vol. 41, pp. 3397–3415, 1993.

[2] Y. Pati, R. Rezaiifar, and P. Krishnaprasad, "Orthogonal matching pursuit: Recursive function approximation with applications to wavelet decomposition," in *27th Annual Asilomar Conference on Signals, Systems, and Computers*, 1993.

[3] S. Chen, D. Donoho, and M. Saunders, "Atomic decomposition by basis pursuit," *SIAM J. Scientific Computing*, vol. 20, no. 1, pp. 33–61, 1999.

[4] I. Drori and D. Donoho, "Solution of L1 minimization problems by LARS/Homotopy methods," in *ICASSP*, 2006, vol. 3, pp. 636–639.

[5] M. Aharon, M. Elad, and A. Bruckstein, "The K-SVD: An algorithm for designing of overcomplete dictionaries for sparse representation," *IEEE Trans. On Signal Processing*, to appear.

[6] J. Starck, M. Elad, and D. Donoho, "Image decomposition via the combination of sparse representation and a variational approach," *IEEE Trans. on Image Processing*, vol. 14, no. 10, pp. 1570–1582, 2005.

[7] Y. Li, A. Cichocki, and S. Amari, "Analysis of sparse representation and blind source separation," *Neural Computation*, vol. 16, no. 6, pp. 1193–1234, 2004.

[8] B. Olshausen, P. Sallee, and M. Lewicki, "Learning sparse image codes using a wavelet pyramid architecture," in *NIPS*, 2001, pp. 887–893.

[9] M. Elad and M. Aharon, "Image denoising via learned dictionaries and sparse representation," in *CVPR*, 2006.

[10] M. Elad, B. Matalon, and M. Zibulevsky, "Image denoising with shrinkage and redundant representation," in *CVPR*, 2006.

[11] D. Donoho and X. Huo, "Uncertainty principles and ideal atomic decomposition," *IEEE Trans. on Information Theory*, vol. 47, no. 7, pp. 2845–2862, 2001.

[12] Y. Lin and D. Lee, "Bayesian L1-Norm sparse learning," in *ICASSP*, 2006, vol. 5, pp. 605–608.

[13] D. Wipf and B. Rao, "Sparse bayesian learning for basis selection," *IEEE Trans. on Signal Processing*, vol. 52, no. 8, pp. 2153–2164, 2004.

[14] R. Duda, P. Hart, and D. Stork, *Pattern classification (2nd ed.)*, Wiley-Interscience, 2000.

[15] P. Belhumeur, J. Hespanha, and D. Kriegman, "Eigenfaces vs. fisherfaces: Recognition using class specific linear projection," *IEEE Transactions on Pattern Analysis and Machine Intelligence*, vol. 19, no. 7, pp. 711–720, 1997.

[16] A. Martinez and A. Kak, "PCA versus LDA," *IEEE Trans. on Pattern Analysis and Machine Intelligence*, vol. 23, no. 2, pp. 228–233, 2001.

[17] S. Fidler, D. Skocaj, and A. Leonardis, "Combining reconstructive and discriminative subspace methods for robust classification and regression by subsampling," *IEEE Trans. on Pattern Analysis and Machine Intelligence*, vol. 28, no. 3, pp. 337–350, 2006.

[18] M. Elad, J. Starck, P. Querre, and D.L. Donoho, "Simultaneous cartoon and texture image inpainting using morphological component analysis (MCA)," *Journal on Applied and Computational Harmonic Analysis*, vol. 19, pp. 340–358, 2005.

[19] A. Leonardis and H. Bischof, "Robust recognition using eigenimages," *Computer Vision and Image Understanding*, vol. 78, pp. 99–118, 2000.

[20] J. Tropp, A. Gilbert, and M. Strauss, "Algorithms for simultaneous sparse approximation. part I: Greedy pursuit," *Signal Processing, special issue on Sparse approximations in signal and image processing*, vol. 86, no. 4, pp. 572–588, 2006.

[21] J. Tropp, A. Gilbert, and M. Strauss, "Algorithms for simultaneous sparse approximation. part II: Convex relaxation," *Signal Processing, special issue on Sparse approximations in signal and image processing*, vol. 86, no. 4, pp. 589–602, 2006.

[22] USPS Handwritten Digit Database, "available at: http://www.cs.toronto.edu/ roweis/data.html," .

[23] R. Kohavi, "A study of cross-validation and bootstrap for accuracy estimation and model selection," in *IJCAI*, 1995, pp. 1137–1145.

[24] M. Tipping, "Sparse bayesian learning and the relevance vector machine," *Journal of Machine Learning Research*, vol. 1, pp. 211–244, 2001.
